# Oscillation Onset
# in
# Neural Delayed Feedback

**André Longtin**
Complex Systems Group and Center for Nonlinear Studies
Theoretical Division B213, Los Alamos National Laboratory
Los Alamos, NM 87545

## Abstract

This paper studies dynamical aspects of neural systems with delayed negative feedback modelled by nonlinear delay-differential equations. These systems undergo a Hopf bifurcation from a stable fixed point to a stable limit cycle oscillation as certain parameters are varied. It is shown that their frequency of oscillation is robust to parameter variations and noisy fluctuations, a property that makes these systems good candidates for pacemakers. The onset of oscillation is postponed by both additive and parametric noise in the sense that the state variable spends more time near the fixed point than it would in the absence of noise. This is also the case when noise affects the delayed variable, i.e. when the system has a faulty memory. Finally, it is shown that a distribution of delays (rather than a fixed delay) also stabilizes the fixed point solution.

## 1  INTRODUCTION

In this paper, we study the dynamics of a class of neural delayed feedback models which have been used to understand equilibrium and oscillatory behavior in recurrent inhibitory circuits (Mackey and an der Heiden, 1984; Plant, 1981; Milton et al., 1990) and brainstem reflexes such as the pupil light reflex (Longtin and Milton, 1989a,b; Milton et al., 1989; Longtin et al., 1990; Longtin, 1991) and respiratory control (Glass and Mackey, 1979). These models are framed in terms of first-order nonlinear delay-differential equations (DDE's) in which the state variable may represent, e.g., a membrane potential, a mean firing rate of a population of neurons or

a muscle activity. For example, the negative feedback dynamics of the human pupil light reflex have been shown to be appropriately modelled by the following equation for pupil area (related to the activity of the iris muscles through the nonlinear monotonically decreasing function $g(A)$ ) (see Longtin and Milton, 1989a,b):

$$\frac{dg(A)}{dA} \frac{dA(t)}{dt} + \alpha g(A) = \gamma \ln \left[ \frac{I(t-\tau)A(t-\tau)}{\overline{\phi}} \right] \tag{1}$$

$I(t)$ is the external light intensity and $\overline{\phi}$ is the retinal light flux below which no pupillary response occurs. The left hand side of Eq.(1) governs the response of the system to the state-dependent forcing (i.e. stimulation) embodied in the term on the right-hand side. The delay $\tau$ is essential to the understanding of the dynamics of this reflex. It accounts for the fact that the iris muscles move in response to the retinal light flux variations occurring $\sim$ 300 msec earlier.

## 2    FOCUS AND MOTIVATION

For the sake of discussion, we shall focus on the following prototypical model of delayed negative feedback

$$\frac{dx(t)}{dt} + \alpha x(t) = f(\vec{\mu}; x(t-\tau)) \tag{2}$$

where $\vec{\mu}$ is a vector of parameters and $f$ is a monotonically decreasing function. This equation typically exhibits a Hopf bifurcation (i.e. a qualitative change in dynamics from a stable equilibrium solution to a stable limit cycle oscillation) as the slope of the feedback function or the delay are increased passed critical values.

Autonomous (as opposed to externally forced) oscillations are frequently observed in real neural delayed feedback systems which suggests that these systems may exhibit a Hopf bifurcation. Further, it is clear that these systems operate despite noisy environmental fluctuations. A clear understanding of the properties of these systems can reveal useful information about their structure and the origin of the "noisy" sources, as well as enable us to extract general functioning principles for systems organized according to this scheme.

We now focus our attention on three different dynamical aspects of these systems: 1) the stability of the oscillation frequency and amplitude to parameter variations and to noise; 2) the postponement of oscillation onset due to noise; and 3) the stabilization of the equilibrium behavior in the more realistic case involving a distribution of delays rather than a single fixed delay.

## 3    FREQUENCY AND AMPLITUDE

Under certain conditions, the neural delayed feedback system will settle onto equilibrium behavior after an initial transient. Mathematically, this corresponds to the fixed point solution $x^*$ of Eq.(2) obtained by setting $\dot{x} = 0$. A supercritical Hopf bifurcation occurs in Eq.(2) when the slope of the feedback function at this fixed point $\frac{df}{dx}\big|_{x^*}$ exceeds some value $k_o$ called the bifurcation value. It can also occur

when the delay exceeds a critical value. The case where the parameter $\alpha$ increases is particularly interesting because the system can undergo a Hopf bifurcation at $\alpha = \alpha_1$ followed by a restabilization of the fixed point through a reverse Hopf bifurcation at $\alpha = \alpha_2 > \alpha_1$ (see also Mackey, 1979).

Numerical simulations of Eq.(2) around the Hopf bifurcation point $k_o$ reveal that the frequency is relatively constant while the amplitude *Ampl* grows as $\sqrt{k - k_o}$. However, in oscillatory time series from real neural delayed feedback systems, the frequency and amplitude fluctuate near the bifurcation point, with relative amplitude fluctuations being generally larger than relative frequency fluctuations. This point has been illustrated using data from the human pupil light reflex whose feedback gain is under experimental control (see Longtin, 1991; Longtin et al., 1990). In the case of the pupil light reflex, the variations in the mean and standard deviation of amplitude and period accompanying increases in the bifurcation parameter (the external gain) have been explained in the hypothesis that "neural noise" is affecting the deterministic dynamics of the system. This noise is strongly amplified near the bifurcation point where the solutions are only weakly stable (Longtin et al., 1990). Thus the coupling of the noise to the system is most likely responsible for the aperiodicity of the observed data.

The fact that the frequency is not significantly affected by the noise nor by variation of the bifurcation parameter (especially in comparison to the amplitude fluctuations) suggests that neural delayed feedback circuits may be ideally suited to serve as pacemakers. The frequency stability in regulatory biological systems has previously been emphasized by Rapp (1981) in the context of biochemical regulation.

## 4   STABILIZATION BY NOISE

In the presence of noise, oscillations can be seen in the solution of Eq.(2) even when the bifurcation value is below that at which the deterministic bifurcation occurs. This does not mean however that the bifurcation has occurred, since these oscillations simply become more and more prominent as the bifurcation parameter is increased, and no qualitative change in the solution can be seen. Such a qualitative change does occur when the solution is viewed from a different standpoint. One can in fact construct a histogram of the values taken on by the solution of the model differential equation (or by the data: see Longtin, 1991). The value of this (normalized) histogram at a given point in the state space (e.g. of pupil area values) provides a measure of the fraction of the time spent by the system in the vicinity of this point. The onset of oscillation can then be detected by a qualitative change in this histogram, specifically when it goes from unimodal to bimodal (Longtin et al., 1990). The distance between the two humps in the bimodal case is a measure of the limit cycle amplitude. For short time series however (as is often the case in neurophysiology), it is practically impossible to resolve this distance and thus to ascertain whether a Hopf bifurcation has occurred.

Intensive simulations of Eq.(2) with either additive noise (i.e. added to Eq.(2)) or parametric noise (e.g. on the magnitude of the feedback function) reveal that the statistical limit cycle amplitude (the distance between the two humps or "order parameter") is smaller than the amplitude in the absence of noise (Longtin et al., 1990). The bifurcation diagram is similar to that in Figure 1. This implies that the

solution spends more time near the fixed point, i.e. that the fixed point is stabilized by the noise (i.e. in the absence of noise, the limit cycle is larger and the system spends less time near the unstable fixed point). In other words, the onset of the Hopf bifurcation is postponed in the presence of these types of noise. Hence the noise level in a neural system, whatever its source, may in fact control the onset of an oscillation.

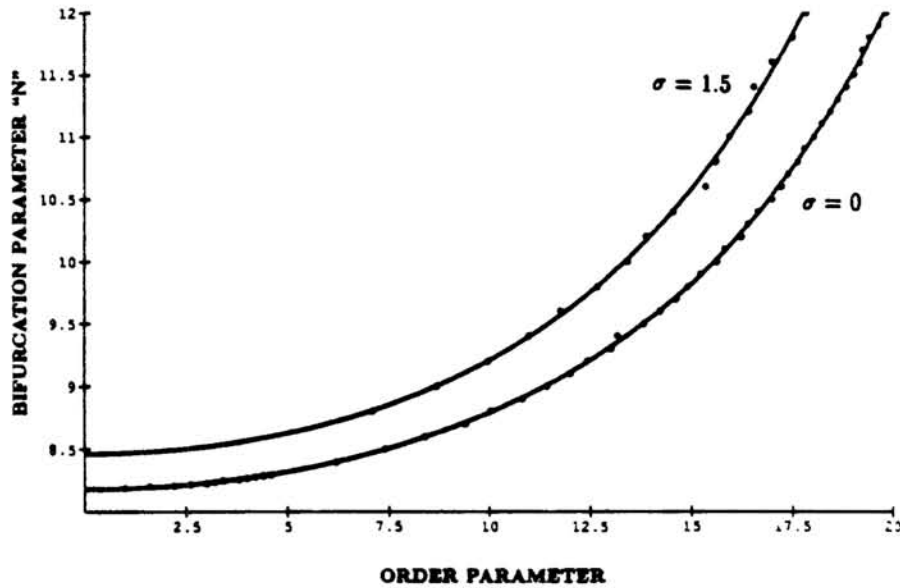

**ORDER PARAMETER**

**Figure 1**. Magnitude of the Order Parameter as a Function of the Bifurcation Parameter $n$ for Noise on the Delayed State of the System.

In Figure 1 it is shown that the Hopf bifurcation is also postponed (the bifurcation curve is shifted to higher parameter values with respect to the deterministic curve) when the noise is applied to the delayed state variable $x(t-\tau)$ and $f$ in Eq.(2) is of the form (negative feedback):

$$f = \frac{\lambda \theta^n}{\theta^n + x^n(t-\tau)}. \tag{3}$$

For parameter values $\alpha = 3.21, \lambda = 200, \theta = 50, \tau = 0.3$, the deterministic Hopf bifurcation occurs at $n = 8.18$. Colored (Ornstein-Uhlenbeck type) Gaussian noise of standard deviation $\sigma = 1.5$ and correlation time $1sec$ was added to the variable $x(t-\tau)$. This numerical calculation can be interpreted as a simulation of the behavior of a neural delayed feedback system with bad memory (i.e. in which there is a small error on the value recalled from the past). Thus, faulty memory also stabilizes the fixed point.

## 5   DISTRIBUTED DELAYS

The use of a single fixed delay in models of delayed feedback is often a good approximation and strongly warranted in a simple circuit comprising only a small number

of cells. However, neural systems often have a spatial extent due to the presence of many parallel pathways in which the axon sizes are distributed according to a certain probability density. This leads to a distribution of conduction velocities down these pathways and therefore to a distribution of propagation delays. In this case, the dynamics are more appropriately modelled by an integro-differential equation of the form

$$\frac{dx}{dt} + \alpha x(t) = f(\vec{\mu}; z(t), x(t)), \quad z(t) = \int_{-\infty}^{t} K(t-u)x(u)\,du. \tag{4}$$

The extent to which values of the state variable in the past affect its present evolution is determined by the kernel $K(t)$. The fixed delay case corresponds to choosing the kernel to be a Dirac delta distribution.

We have looked at the effect of a distributed delay on the Hopf bifurcation in our prototypical delayed feedback system Eq.(2). Specifically, we have considered the case where the kernel in Eq.(4) has the form of a gamma distribution

$$K(t) \equiv G_a^m(t) = \frac{a^{m+1}}{m!} t^m e^{-aq}, \quad a, m \geq 0. \tag{5}$$

The average delay of this kernel is $\bar{\tau} = \frac{m+1}{a}$ and the kernel has the property that it converges to the delta function in the limit where $m$ and $a$ go to infinity all the while keeping the ratio $\bar{\tau}$ constant. For a kernel of a given order it is possible to convert the DDE Eq.(2) into a set of $(m+2)$ coupled ordinary differential equations (ODE's) which approximate the DDE (an infinite set of ODE's is in this case equivalent to the original DDE) (see Fargue, 1973; MacDonald, 1978; Cooke and Grossman, 1982). We have investigated the occurrence of a Hopf bifurcation in the $(m+2)$ ODE's as a function of the order $m$ of the memory kernel (keeping $\bar{\tau}$ equal to the fixed delay of the DDE being approximated). This involves doing a stability analysis around the fixed point of the $(m+2)$ order system of ODE's and numerically determining the value of the bifurcation parameter $n$ at which the Hopf bifurcation occurs.

The result is shown in Figure 2, where we have plotted $n$ versus the order $m$ of approximation. Note that at least a 3 dimensional system of ODE's is required for a Hopf bifurcation to occur in such a system. Note also the fast convergence of $n$ to the bifurcation value for the DDE (5.04). These calculations were done for the Mackey-Glass equation

$$\frac{dx}{dt} + \alpha x(t) = \frac{\lambda \theta^n x(t-\tau)}{\theta^n + x^n(t-\tau)} \tag{6}$$

with parameters $\theta = 1, \alpha = 2, \lambda = 2, \tau = 2$ and $n \in (1, 20)$. This equation is a model for mixed feedback dynamics (i.e. a combination of positive and negative feedback involving a single-humped feedback function). It displays the same qualitative features as Eq.(2) with the feedback given by Eq.(3) at the Hopf bifurcation and was chosen for ease of computation since parameters can be chosen such that the fixed point does not depend on the bifurcation parameter.

We can see that, for a memory kernel of a given order, the Hopf bifurcation occurs at a higher value of the bifurcation parameter (which is proportional to the slope of the feedback function at the fixed point) than for the DDE. This implies that a stronger nonlinearity is required to set the ODE system into oscillation compared

to the DDE. In other words, the distributed delay system with the same feedback function as the DDE is less prone to oscillate (see also MacDonald, 1978; Cooke and Grossman, 1982).

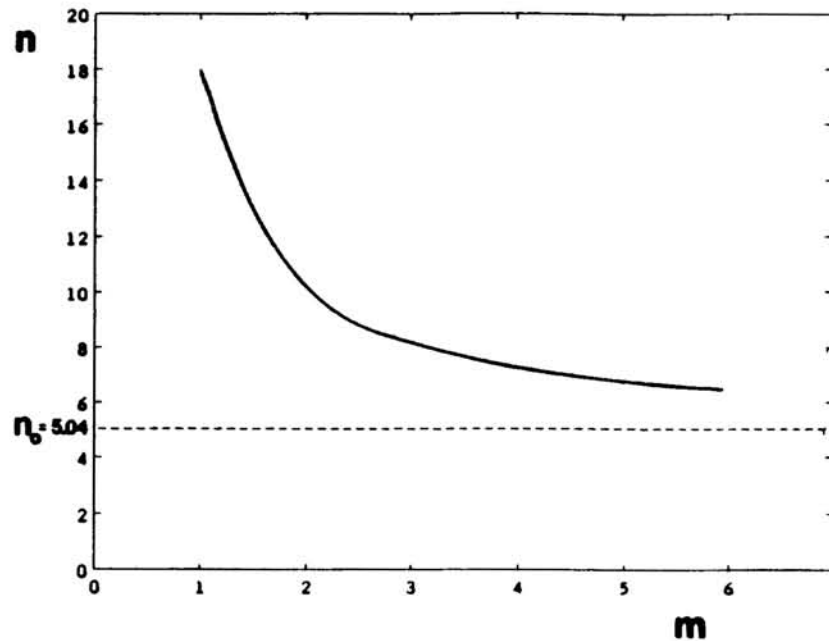

**Figure 2.** Value of $n$ at Which a Hopf Bifurcation Occurs Versus the Order $m$ of the Memory Kernel.

# 6  SUMMARY

In summary we have shown that neural delayed negative feedback systems can exhibit either equilibrium or limit cycle behavior depending on their parameters and on the noise levels. The constancy of their oscillation frequency, even in the presence of noise, suggests their possible role as pacemakers in the nervous system. Further, the equilibrium solution of these systems is stabilized by noise and by distributed delays. We conjecture that these two effects may be related as they somewhat share a common feature, in the sense that noise and distributed delays tend to make the retarded action more diffuse. This is supported by the fact that a system with bad memory (i.e. with noise on the delayed variable) also sees its fixed point stabilized.

**Acknowledgements**

The author would like to thank Mackey for useful conversations as well as Christian Cortis for his help with the numerical analysis in Section 5. This research was supported by the Natural Sciences and Engineering Research Council of Canada (NSERC) as well as the Complex Systems Group and the Center for Nonlinear Studies at Los Alamos National Laboratory in the form of postdoctoral fellowships.

## References

K.L. Cooke and Z. Grossman. (1982) Discrete delay, distributed delay and stability switches. *J. Math. Anal. Appl.* **86**:592-627.

D. Fargue. (1973) Réductibilité des systèmes héréditaires a des systèmes dynamiques (régis par des équations différentielles aux dérivées partielles). *C.R. Acad. Sci. Paris* **T.277, No.17** (Serie B, 2e semestre):471-473.

L. Glass and M.C. Mackey. (1979) Pathological conditions resulting from instabilities in physiological control systems. *Ann. N.Y. Acad. Sci.* **316**:214.

A. Longtin. (in press, 1991) Nonlinear dynamics of neural delayed feedback. In D. Stein (ed.),*Proceedings of the $3^{rd}$ Summer School on Complex Systems, Santa Fe Institute Studies in the Sciences of Complexity, Lect. Vol. III.* Redwood City, CA: Addison-Wesley.

A. Longtin and J.G. Milton. (1989a) Modelling autonomous oscillations in the human pupil light reflex using nonlinear delay-differential equations. *Bull. Math. Biol.* **51**:605-624.

A. Longtin and J.G. Milton. (1989b) Insight into the transfer function, gain and oscillation onset for the pupil light reflex using nonlinear delay-differential equations. *Biol. Cybern.* **61**:51-59.

A. Longtin, J.G. Milton, J. Bos and M.C. Mackey. (1990) Noise and critical behavior of the pupil light reflex at oscillation onset. *Phys. Rev.* **A 41**:6992-7005.

N. MacDonald. (1978) Time lags in biological models. *Lecture Notes in Biomathematics* **27**. Berlin: Springer Verlag.

M.C. Mackey. (1979) Periodic auto-immune hemolytic anemia: an induced dynamical disease. *Bull. Math. Biol.* **41**:829-834.

M.C. Mackey and U. an der Heiden. (1984) The dynamics of recurrent inhibition. *J. Math. Biol.* **19**: 211-225.

J.G. Milton, U. an der Heiden, A. Longtin and M.C. Mackey. (in press, 1990) Complex dynamics and noise in simple neural networks with delayed mixed feedback. *Biomed. Biochem. Acta* **8/9**.

J.G. Milton, A. Longtin, A. Beuter, M.C. Mackey and L. Glass. (1989) Complex dynamics and bifurcations in neurology. *J. Theor. Biol.* **138**:129-147.

R.E. Plant. (1981) A Fitzhugh differential-difference equation modelling recurrent neural feedback. *SIAM J. Appl. Math.* **40**:150-162.

P.E. Rapp. (1981) Frequency encoded biochemical regulation is more accurate then amplitude dependent control. *J. Theor. Biol.* **90**:531-544.